# Statistical Mechanics of Learning in a Large Committee Machine

**Holm Schwarze**
CONNECT, The Niels Bohr Institute
Blegdamsvej 17, DK-2100 Copenhagen Ø, Denmark

**John Hertz***
Nordita
Blegdamsvej 17, DK-2100 Copenhagen Ø, Denmark

## Abstract

We use statistical mechanics to study generalization in large committee machines. For an architecture with nonoverlapping receptive fields a replica calculation yields the generalization error in the limit of a large number of hidden units. For continuous weights the generalization error falls off asymptotically inversely proportional to $\alpha$, the number of training examples per weight. For binary weights we find a discontinuous transition from poor to perfect generalization followed by a wide region of metastability. Broken replica symmetry is found within this region at low temperatures. For a fully connected architecture the generalization error is calculated within the annealed approximation. For both binary and continuous weights we find transitions from a symmetric state to one with specialized hidden units, accompanied by discontinuous drops in the generalization error.

## 1   Introduction

There has been a good deal of theoretical work on calculating the *generalization ability* of neural networks within the framework of statistical mechanics (for a review

see e.g. Watkin et.al., 1992; Seung et.al., 1992). This approach has mostly been applied to single–layer nets (e.g. Györgyi and Tishby, 1990; Seung et.al., 1992). Extensions to networks with a hidden layer include a model with small hidden receptive fields (Sompolinsky and Tishby, 1990), some general results on networks whose outputs are continuous functions of their inputs (Seung et.al., 1992; Krogh and Hertz, 1992), and calculations for a so–called *committee machine* (Nilsson, 1965), a two–layer Boolean network, which implements a majority decision of the hidden units (Schwarze et.al., 1992; Schwarze and Hertz, 1992; Mato and Parga, 1992; Barkai et.al., 1992; Engel et.al., 1992). This model has previously been studied when learning a function which could be implemented by a simple perceptron (i.e. one with no hidden units) in the high–temperature (i.e. high–noise) limit (Schwarze et.al., 1992). In most practical applications, however, the function to be learnt is not linearly separable. Therefore, we consider here a committee machine trained on a rule which itself is defined by another committee machine (the 'teacher' network) and hence not linearly separable.

We calculate the generalization error, the probability of misclassifying an arbitrary new input, as a function of $\alpha$, the ratio of the number of training examples $P$ to the number of adjustable weights in the network. First we present results for the 'tree' committee machine, a restricted version of the model in which the receptive fields of the hidden units do not overlap. In section 3 we study a fully connected architecture allowing for correlations between different hidden units in the student network. In both cases we study a large–net limit in which the total number of inputs ($N$) and the number of hidden units ($K$) both go to infinity, but with $K \ll N$.

## 2   Committee machine with nonoverlapping receptive fields

In this model each hidden unit receives its input from $N/K$ input units, subject to the restriction that different hidden units do not share common inputs. Therefore there is only one path from each input unit to the output. The hidden–output weights are all fixed to $+1$ as to implement a majority decision of the hidden units. The overall network output for inputs $\underline{S}_l \in \mathbb{R}^{N/K}$, $l = 1, \ldots, K$, to the $K$ branches is given by

$$\sigma(\{\underline{S}_l\}) = \text{sign}\left(\frac{1}{\sqrt{K}} \sum_{l=1}^{K} \sigma_l\left(\underline{S}_l\right)\right), \tag{1}$$

where $\sigma_l$ is the output of the $l$th hidden unit, given by

$$\sigma_l(\underline{S}_l) = \text{sign}\left(\sqrt{\frac{K}{N}} \underline{W}_l \cdot \underline{S}_l\right). \tag{2}$$

Here $\underline{W}_l$ is the $N/K$–dimensional weight vector connecting the input with the $l$th hidden unit. The training examples $(\{\underline{\xi}^\mu_l\}, \tau(\{\underline{\xi}^\mu_l\})), \mu = 1, \ldots, P$, are generated by another committee machine with weight vectors $\underline{V}_l$ and an overall output $\tau(\{\underline{\xi}^\mu_l\})$, defined analogously to (1). There are $N$ adjustable weights in the network, and therefore we have $\alpha = P/N$.

As in the corresponding calculations for simple perceptrons (Gardner and Derrida, 1988; Györgyi and Tishby, 1990; Seung et.al., 1992), we consider a stochastic learning algorithm which for long training times yields a Gibbs distribution of networks. The statistical mechanics approach starts out from the partition function $Z = \int d\rho_0(\{\underline{W}_l\}) \, e^{-\beta E(\{\underline{w}_l\})}$, an integral over weight space with a priori measure $\rho_0(\{\underline{W}_l\})$, weighted with a thermal factor $e^{-\beta E(\{\underline{w}_l\})}$, where $E$ is the total error on the training examples

$$E(\{\underline{W}_l\}) = \sum_{\mu=1}^{P} \Theta\left[-\sigma\left(\{\underline{\xi}_l^\mu\}\right) \cdot \tau\left(\{\underline{\xi}_l^\mu\}\right)\right]. \tag{3}$$

The formal temperature $T = 1/\beta$ defines the level of noise during the training process. For $T = 0$ this procedure corresponds to simply minimizing the training error $E$.

From this the average free energy $F = -T \langle\!\langle \ln Z \rangle\!\rangle$, averaged over all possible sets of training examples can be calculated using the replica method (for details see Schwarze and Hertz, 1992). Like the calculations for simple perceptrons, our theory has two sets of order parameters:

$$q_l^{\alpha\beta} = \frac{K}{N} \, \underline{W}_l^\alpha \cdot \underline{W}_l^\beta \qquad R_l^\alpha = \frac{K}{N} \, \underline{W}_l^\alpha \cdot \underline{V}_l.$$

Note that these are the only order parameters in this model. Due to the tree structure no correlations between different hidden units exist. Assuming both replica symmetry and 'translational symmetry' we are left with two parameters: $q$, the pattern average of the square of the average input–hidden weight vector, and $R$, the average overlap between this weight vector and a corresponding one for the teacher.

We then obtain expressions for the replica–symmetric free energy of the form $G(q, R, \hat{q}, \hat{R}) = \alpha \, G_1(q, R) + G_2(q, R, \hat{q}, \hat{R})$, where the 'entropy' terms $G_2$ for the continuous– and binary–weight cases are exactly the same as in the simple perceptron (Györgyi and Tishby, 1990, Seung et.al., 1992). In the large–$K$ limit another simplification similar to the zero–temperature capacity calculation (Barkai et.al., 1992) is found in the tree model. The 'energy' term $G_1$ is the same as the corresponding term in the calculation for the simple perceptron, except that the order parameters have to be replaced by $f(q) = (2/\pi)\sin^{-1} q$ and $f(R) = (2/\pi)\sin^{-1} R$. The generalization error

$$\epsilon_g = \frac{1}{\pi} \arccos\left[f(R)\right] \tag{4}$$

can then be obtained from the value of $R$ at the saddle point of the free energy.

For a network with continuous weights, the solution of the saddle point equations yields an algebraically decreasing generalization error. There is no phase transition at any value of $\alpha$ or $T$. For $T = 0$ the asymptotic form of the generalization error in powers of $1/\alpha$ can be easily obtained as $1.25/\alpha + \mathcal{O}(1/\alpha^2)$, twice the $\epsilon_g$ found for the simple perceptron in this limit.

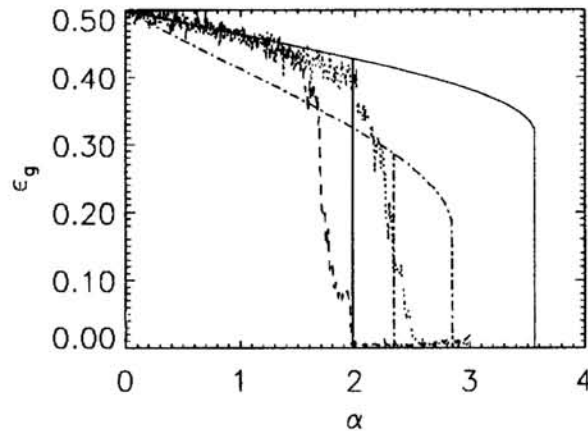

Figure 1: Learning curve for the large-$K$ tree committee (solid line) with binary weights at $T = 1$. The phase transition occurs at $\alpha_c = 1.98$, and the spinodal point is at $\alpha_s = 3.56$. The analytic results are compared with Monte Carlo simulations with $K = 9$, $N = 75$ and $T = 1$, averaged over 10 runs. In each simulation the number of training examples is gradually increased (dotted line) and decreased (dashed line), respectively. The broken line shows the generalization error for the simple perceptron.

In contrast, the model with binary weights exhibits a phase transition at all temperatures from poor to perfect generalization. The corresponding generalization error as a function of $\alpha$ is shown in figure 1. At small values of $\alpha$ the free energy has two saddle points, one at $R < 1$ and the other at $R = 1$. Initially the solution with $R < 1$ and poor generalization ability has the lower free energy and therefore corresponds to the equilibrium state. When the load parameter is increased to a critical value $\alpha_c$, the situation changes and the solution at $R = 1$ becomes the global minimum of the free energy. The system exhibits a first order phase transition to the state of perfect generalization. In the region $\alpha_c < \alpha < \alpha_s$ the $R < 1$ solution remains metastable and disappears at the spinodal point $\alpha_s$. We find the same qualitative picture at all temperatures, and the complete replica symmetric phase diagram is shown in figure 2. The solid line corresponds to the phase transition to perfect generalization, and in the region between the solid and the dashed lines the $R < 1$ state of poor generalization is metastable. Below the dotted line, the replica–symmetric solution yields a negative entropy for the metastable state. This is unphysical in a binary system and replica symmetry has to be broken in this region, indicating the existence of many different metastable states.

The simple perceptron without hidden units corresponds to the case $K = 1$ in our model. A comparison of the generalization properties with the large-$K$ limit shows that both limits exhibit qualitatively similar behavior. The locations of the thermodynamic transitions and the spinodal line, however, are different and the generalization error of the $R < 1$ state in the large-$K$ committee machine is higher than in the simple perceptron.

The case of general finite $K$ is rather more involved, but the annealed approximation

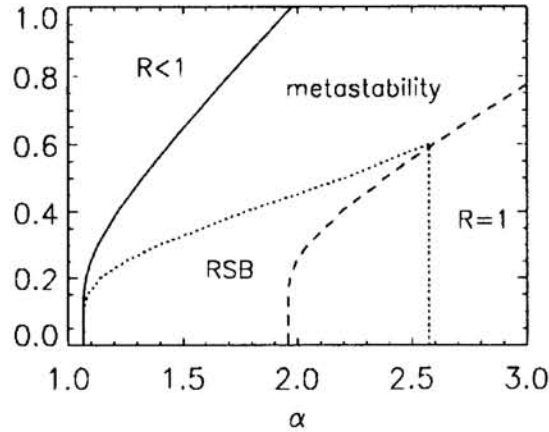

Figure 2: Replica–symmetric phase diagram of the large–$K$ tree committee machine with binary weights. The solid line shows the locations of the phase transition, and the spinodal line is shown dashed. Below the the dotted line the replica–symmetric solution is incorrect.

for finite $K$ indicates a rather smooth $K$–dependence for $1 < K < \infty$ (Mato and Parga, 1992).

We performed Monte–Carlo simulations to check the validity of the assumptions made in our calculation and found good agreements with our analytic results. Figure 1 compares the analytic predictions for large $K$ with Monte Carlo simulations for $K = 9$. The simulations were performed for a slowly increasing and decreasing training set size, respectively, yielding a hysteresis loop around the location of the phase transition.

## 3   Fully connected committee machine

In contrast to the previous model the hidden units in the fully connected committee machine receive inputs from the entire input layer. Their output for a given $N$–dimensional input vector $\underline{S}$ is given by

$$\sigma_l(\underline{S}) = \text{sign}\left(\frac{1}{\sqrt{N}}\underline{W}_l \cdot \underline{S}\right),\tag{5}$$

while the overall output is again of the form (1). Note that the weight vectors $\underline{W}_l$ are now $N$–dimensional, and the load parameter is given by $\alpha = P/(KN)$.

For this model we solved the annealed approximation, which replaces $\langle\!\langle \ln Z \rangle\!\rangle$ by $\ln \langle\!\langle Z \rangle\!\rangle$. This approximation becomes exact at high temperatures (high noise level during training). For learnable target rules, as in the present problem, previous work indicates that the annealed approximation yields qualitatively correct results and correctly predicts the shape of the learning curves even at low temperatures (Seung et.al., 1992). Performing the average over all possible training sets again leads to two sets of order parameters: the overlaps between the student and teacher weight

vectors, $R_{lk} = N^{-1} \underline{W}_l \cdot \underline{V}_k$, and the mutual overlaps in the student network $C_{lk} = N^{-1} \underline{W}_l \cdot \underline{W}_k$. The weight vectors of the target rule are assumed to be uncorrelated and normalized, $N^{-1} \underline{V}_l \cdot \underline{V}_k = \delta_{lk}$. As in the previous model we make symmetry assumptions for the order parameters. In the fully connected architecture we have to allow for correlations between different hidden units ($R_{lk}, C_{lk} \neq 0$ for $l \neq k$) but also include the possibility of a specialization of individual units ($R_{ll} \neq R_{lk}$). This is necessary because the ground state of the system with vanishing generalization error is achieved for the choice $R_{lk} = C_{lk} = \delta_{lk}$. Therefore we make the ansatz

$$R_{lk} = R + \Delta \delta_{lk}, \qquad C_{lk} = C + (1 - C)\delta_{lk} \tag{6}$$

and evaluate the annealed free energy of the system using the saddle point method (details will be reported elsewhere). The values of the order parameters at the minimum of the free energy finally yield the average generalization error $\epsilon_g$ as a function of $\alpha$.

For a network with continuous weights and small $\alpha$ the global minimum of the free energy occurs at $\Delta = 0$ and $R \sim \mathcal{O}(K^{-3/4})$. Hence, for small training sets each hidden unit in the student network has a small symmetric overlap to all the hidden units in the teacher network. The information obtained from the training examples is not sufficient for a specialization of hidden units, and the generalization error approaches a plateau. To order $1/\sqrt{K}$, this approach is given by

$$\epsilon_g = \epsilon_0 + \sqrt{\frac{\gamma(\beta)}{\alpha K}} + \mathcal{O}(1/K), \quad \epsilon_0 = \frac{1}{\pi} \arccos\left(\sqrt{2/\pi}\right) \approx 0.206, \tag{7}$$

with $\gamma(\beta) = \sqrt{\pi/2 - 1}\,[(1 - e^{-\beta})^{-1} - \epsilon_0]/(4\pi)$. Figure 3 shows the generalization error as a function of $\alpha$, including $1/\sqrt{K}$–corrections for different values of $K$.

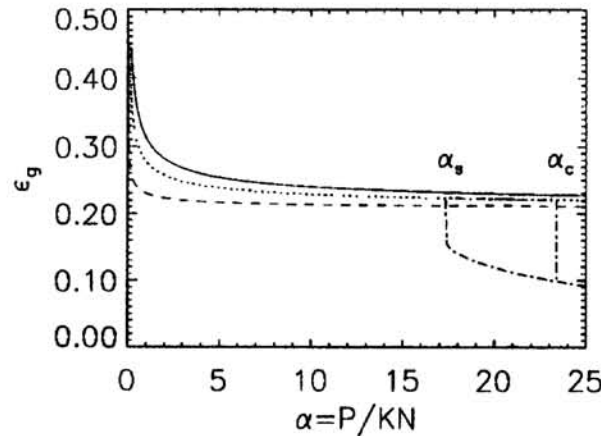

Figure 3: Generalization error for continuous weights and $T = 0.5$. The approach to the residual error is shown including $1/\sqrt{K}$–corrections for K=5 (solid line), K=11 (dotted line), and K=100 (dashed line). The broken line corresponds to the solution with nonvanishing $\Delta$.

When the training set size is increased to a critical value $\alpha_s$ of the load parameter,

a second minimum of the free energy appears at a finite value of $\Delta$ close to 1. For a larger value $\alpha_c > \alpha_s$ this becomes the global minimum of the free energy and the system exhibits a first order phase transition. The generalization error of the specialized solution decays smoothly with an asymptotic behavior inversely proportional to $\alpha$. However, the poorly-generalizing symmetric state remains metastable for all $\alpha > \alpha_c$. Therefore, a stochastic learning procedure starting with $\Delta = 0$ will first settle into the metastable state. For large $N$ it will take an exponentially long time to cross the free energy barrier to the global minimum of the free energy.

In a network with binary weights and for large $K$ we find the same initial approach to a finite generalization error as in (7) for continuous weights. In the large–$K$ limit the discreteness of the weights does not influence the behavior for small training sets. However, while a perfect match of the student to the teacher network ($R_{lk} = C_{lk} = \delta_{lk}$) cannot happen for $\alpha < \infty$ in the continuous model, such a 'freezing' is possible in a discrete system. The free energy of the binary model always has a local minimum at $R_{lk} = C_{lk} = \delta_{lk}$. When the load parameter is increased to a critical value, this minimum becomes the global minimum of the free energy, and a discontinuous transition into this perfectly generalizing state occurs, just as in the binary–weight simple perceptron and the tree described in section 2. As in the case of continuous weights, the symmetric solution remains metastable here even for large values of $\alpha$. Figure 4 shows the generalization error for binary weights, including $1/\sqrt{K}$–corrections for $K = 5$. The predictions of the large–$K$ theory are compared with Monte Carlo simulations. Although we cannot expect a good quantitative agreement for such a small committee, the simulations support our qualitative results. Note that the leading order correction to $\epsilon_0$ in eqn. (7) is only small for $\alpha \gg 1/K$. However, we have obtained a different solution, which is valid for $\alpha \sim \mathcal{O}(1/K)$. The corresponding generalization error is shown as a dotted line in figure 4.

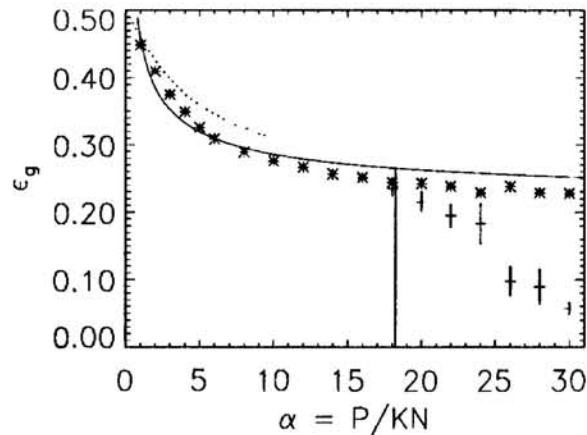

Figure 4: Generalization error for binary weights at $T = 5$. The large–$K$ theory for different regions of $\alpha$ is compared with simulations for $K = 5$ and $N = 45$ averaged over all simulations (+) and simulations, in which no freezing occurred (*), respectively. The solid line shows the finite–$\alpha$ results including $1/\sqrt{K}$–corrections. The dotted line shows the small–$\alpha$ solution.

Compared to the tree model the fully connected committee machine shows a qualitatively different behavior. This difference is particularly pronounced in the continuous model. While the generalization error of the tree architecture decays smoothly for all values of $\alpha$, the fully connected model exhibits a discontinuous phase transition. Compared to the tree model, the fully connected architecture has an additional symmetry, because each permutation of hidden units in the student network yields the same output for a given input (Barkai et.al., 1992). This additional degree of freedom causes the poor generalization ability for small training sets. Only if the training set size is sufficiently large can the hidden units specialize on one of the hidden units in the teacher network and achieve good generalization. However, the poorly generalizing states remain metastable even for arbitrarily large $\alpha$. A similar phenomenon has also been found in a different architecture with only 2 hidden units performing a parity operation (Hansel et.al., 1992).

## Acknowledgements

H. Schwarze acknowledges support from the EC under the SCIENCE programme and by the Danish Natural Science Council and the Danish Technical Research Council through CONNECT.

## Footnotes

*Address in 1993: Laboratory of Neuropsychology, NIMH, Bethesda, MD 20892, USA

## References

E. Barkai, D. Hansel, and H. Sompolinsky (1992), Phys.Rev. A **45**, 4146.

A. Engel, H.M. Köhler, F. Tschepke, H. Vollmayr, and A. Zippelius (1992), Phys.Rev. A **45**, 7590.

E. Gardner, B. Derrida (1989), J.Phys. A **21**, 271.

G. Györgyi and N. Tishby (1990) in *Neural Networks and Spin Glasses*, edited K. Thuemann and R. Köberle (World Scientific, Singapore).

D. Hansel, G. Mato, and C. Meunier (1992), Europhys.Lett. **20**, 471.

A. Krogh, J. Hertz (1992), *Advances in Neural Information Processing Systems IV*, edited by J.E. Moody, S.J. Hanson, and R.P. Lippmann, (Morgan Kaufmann, San Mateo).

G. Mato, N. Parga (1992), J.Phys. A **25**, 5047.

N.J. Nilsson (1965) *Learning Machines*, (McGraw–Hill, New York).

H. Schwarze, M. Opper, and W. Kinzel (1992), Phys.Rev. A **45**, R6185.

H. Schwarze, J. Hertz (1992), Europhys.Lett. **20**, 375.

H.S. Seung, H. Sompolinsky, and N. Tishby (1992), Phys.Rev. A **45**, 6056.

H. Sompolinsky, N. Tishby (1990), Europhys.Lett. **13**, 567.

T. Watkin, A. Rau, and M. Biehl (1992), to be published in Review of Modern Physics.